# Continuous Speech Recognition by Linked Predictive Neural Networks

**Joe Tebelskis, Alex Waibel, Bojan Petek, and Otto Schmidbauer**
School of Computer Science
Carnegie Mellon University
Pittsburgh, PA 15213

## Abstract

We present a large vocabulary, continuous speech recognition system based on Linked Predictive Neural Networks (LPNN's). The system uses neural networks as predictors of speech frames, yielding distortion measures which are used by the One Stage DTW algorithm to perform continuous speech recognition. The system, already deployed in a Speech to Speech Translation system, currently achieves 95%, 58%, and 39% word accuracy on tasks with perplexity 5, 111, and 402 respectively, outperforming several simple HMMs that we tested. We also found that the accuracy and speed of the LPNN can be slightly improved by the judicious use of hidden control inputs. We conclude by discussing the strengths and weaknesses of the predictive approach.

## 1 INTRODUCTION

Neural networks are proving to be useful for difficult tasks such as speech recognition, because they can easily be trained to compute smooth, nonlinear, nonparametric functions from any input space to output space. In speech recognition, the function most often computed by networks is *classification*, in which spectral frames are mapped into a finite set of classes, such as phonemes. In theory, classification networks approximate the optimal Bayesian discriminant function [1], and in practice they have yielded very high accuracy [2, 3, 4]. However, integrating a phoneme classifier into a speech recognition system is nontrivial, since classification decisions tend to be binary, and binary phoneme-level errors tend to confound word-level hypotheses. To circumvent this problem, neural network training must be carefully integrated into word level training [1, 5]. An alternative function which can be com-

puted by networks is *prediction*, where spectral frames are mapped into predicted spectral frames. This provides a simple way to get non-binary distortion measures, with straightforward integration into a speech recognition system. Predictive networks have been used successfully for small vocabulary [6, 7] and large vocabulary [8, 9] speech recognition systems. In this paper we describe our prediction-based LPNN system [9], which performs large vocabulary continuous speech recognition, and which has already been deployed within a Speech to Speech Translation system [10]. We present our experimental results, and discuss the strengths and weaknesses of the predictive approach.

## 2   LINKED PREDICTIVE NEURAL NETWORKS

The LPNN system is based on canonical phoneme models, which can be logically concatenated in any order (using a "linkage pattern") to create templates for different words; this makes the LPNN suitable for large vocabulary recognition.

Each canonical phoneme is modeled by a short sequence of neural networks. The number of nets in the sequence, $N >= 1$, corresponds to the granularity of the phoneme model. These phone modeling networks are nonlinear, multilayered, feedforward, and "predictive" in the sense that, given a short section of speech, the networks are required to extrapolate the raw speech signal, rather than to classify it. Thus, each predictive network produces a time-varying model of the speech signal which will be accurate in regions corresponding to the phoneme for which that network has been trained, but inaccurate in other regions (which are better modeled by other networks). Phonemes are thus "recognized" indirectly, by virtue of the relative accuracies of the different predictive networks in various sections of speech. Note, however, that phonemes are not classified at the frame level. Instead, continuous scores (prediction errors) are accumulated for various word candidates, and a decision is made only at the word level, where it is finally appropriate.

### 2.1   TRAINING AND TESTING ALGORITHMS

The purpose of the training procedure is both (a) to train the networks to become better predictors, and (b) to cause the networks to specialize on different phonemes. Given a known training utterance, the training procedure consists of three steps:

1. Forward Pass: All the networks make their predictions across the speech sample, and we compute the Euclidean distance matrix of prediction errors between predicted and actual speech frames. (See Figure 1.)
2. Alignment Step: We compute the optimal time-alignment path between the input speech and corresponding predictor nets, using Dynamic Time Warping.
3. Backward Pass: Prediction error is backpropagated into the networks according to the segmentation given by the alignment path. (See Figure 2.)

Hence backpropagation causes the nets to become better predictors, and the alignment path induces specialization of the networks for different phonemes.

Testing is performed using the One Stage algorithm [11], which is a classical extension of the Dynamic Time Warping algorithm for continuous speech.

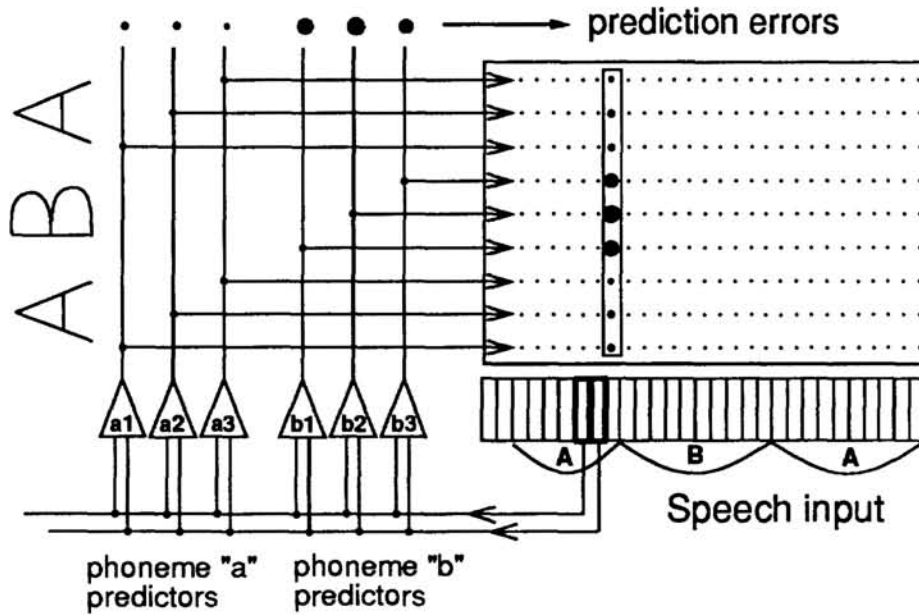

Figure 1: The forward pass during training. Canonical phonemes are modeled by sequences of N predictive networks, shown as triangles (here N=3). Words are represented by "linkage patterns" over these canonical phoneme models (shown in the area above the triangles), according to the phonetic spelling of the words. Here we are training on the word "ABA". In the forward pass, prediction errors (shown as black circles) are computed for all predictors, for each frame of the input speech. As these prediction errors are routed through the linkage pattern, they fill a distance matrix (upper right).

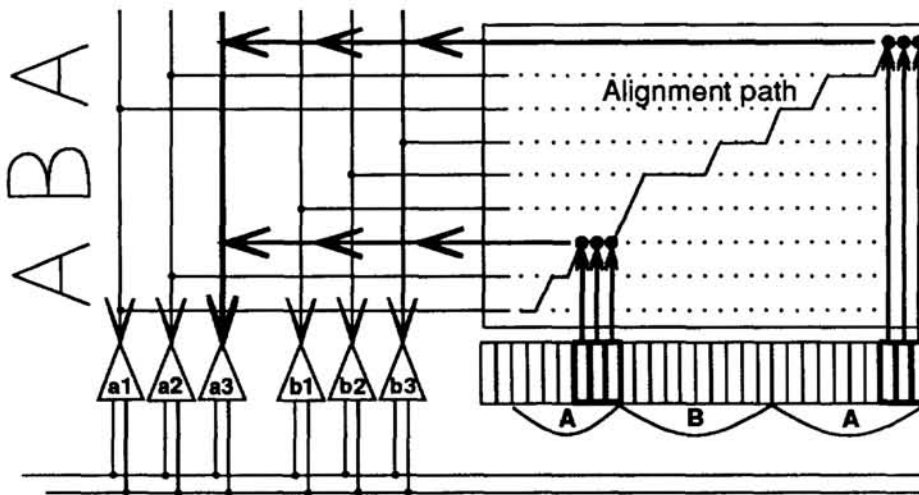

Figure 2: The backward pass during training. After the DTW alignment path has been computed, error is backpropagated into the various predictors responsible for each point along the alignment path. The backpropagated error signal at each such point is the vector difference between the predicted and actual frame. This teaches the networks to become better predictors, and also causes the networks to specialize on different phonemes.

# 3   RECOGNITION EXPERIMENTS

We have evaluated the LPNN system on a database of continuous speech recorded at CMU. The database consists of 204 English sentences using a vocabulary of 402 words, comprising 12 dialogs in the domain of conference registration. Training and testing versions of this database were recorded in a quiet office by multiple speakers for speaker-dependent experiments. Recordings were digitized at a sampling rate of 16 KHz. A Hamming window and an FFT were computed, to produce 16 melscale spectral coefficients every 10 msec. In our experiments we used 40 context-independent phoneme models (including one for silence), each of which had a 6-state phoneme topology similar to the one used in the SPICOS system [12].

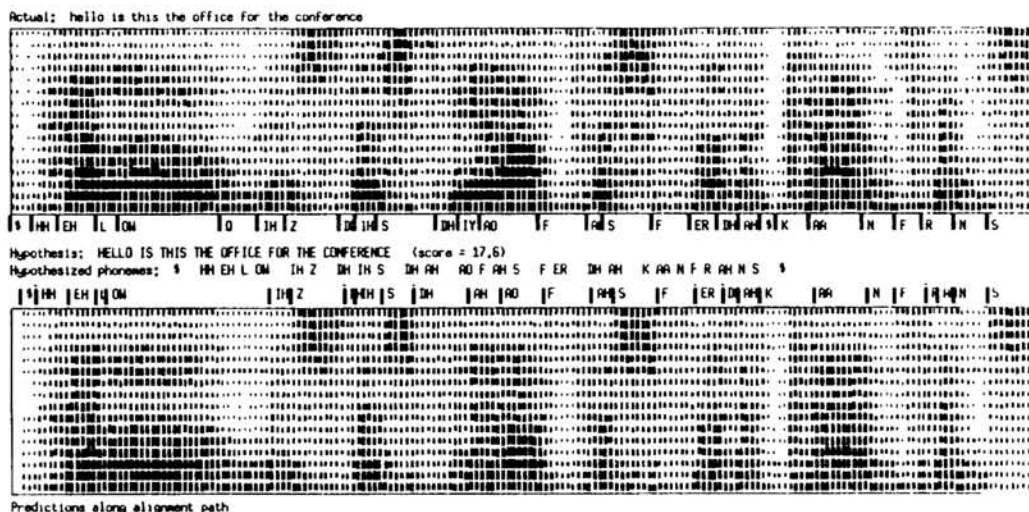

Figure 3: Actual and predicted spectrograms.

Figure 3 shows the result of testing the LPNN system on a typical sentence. The top portion is the actual spectrogram for this utterance; the bottom portion shows the frame-by-frame predictions made by the networks specified by each point along the optimal alignment path. The similarity of these two spectrograms indicates that the hypothesis forms a good acoustic model of the unknown utterance (in fact the hypothesis was correct in this case). In our speaker-dependent experiments using two males speakers, our system averaged 95%, 58%, and 39% word accuracy on tasks with perplexity 5, 111, and 402 respectively.

In order to confirm that the predictive networks were making a positive contribution to the overall system, we performed a set of comparisons between the LPNN and several pure HMM systems. When we replaced each predictive network by a univariate Gaussian whose mean and variance were determined analytically from the labeled training data, the resulting HMM achieved 44% word accuracy, compared to 60% achieved by the LPNN under the same conditions (single speaker, perplexity 111). When we also provided the HMM with delta coefficients (which were not directly available to the LPNN), it achieved 55%. Thus the LPNN was outperforming each of these simple HMMs.

## 4    HIDDEN CONTROL EXPERIMENTS

In another series of experiments, we varied the LPNN architecture by introducing hidden control inputs, as proposed by Levin [7]. The idea, illustrated in Figure 4, is that a sequence of independent networks is replaced by a single network which is modulated by an equivalent number of "hidden control" input bits that distinguish the state.

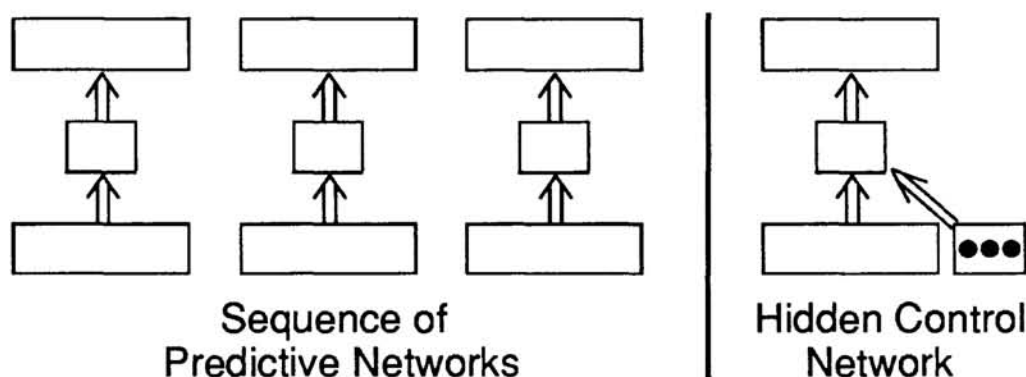

Sequence of
Predictive Networks

Hidden Control
Network

Figure 4: A sequence of networks corresponds to a single Hidden Control network.

A theoretical advantage of hidden control architectures is that they reduce the number of free parameters in the system. As the number of networks is reduced, each one is exposed to more training data, and – up to a certain point – generalization may improve. The system can also run faster, since partial results of redundant forward pass computations can be saved. (Notice, however, that the total number of forward passes is unchanged.) Finally, the savings in memory can be significant.

In our experiments, we found that by replacing 2-state phoneme models by equivalent Hidden Control networks, recognition accuracy improved slightly and the system ran much faster. On the other hand, when we replaced *all* of the phonemic networks in the entire system by a single Hidden Control network (whose hidden control inputs represented the phoneme as well as its state), recognition accuracy degraded significantly. Hence, hidden control may be useful, but only if it is used judiciously.

## 5    CURRENT LIMITATIONS OF PREDICTIVE NETS

While the LPNN system is good at modeling the acoustics of speech, it presently tends to suffer from poor discrimination. In other words, for a given segment of speech, all of the phoneme models tend to make similarly good predictions, rendering all phoneme models fairly confusable. For example, Figure 5 shows an actual spectrogram and the frame-by-frame predictions made by the /eh/ model and the /z/ model. Disappointingly, both models are fairly accurate predictors for the entire utterance.

This problem arises because each predictor receives training in only a small region of input acoustic space (i.e., those frames corresponding to that phoneme). Consequently, when a predictor is shown any other input frames, it will compute an

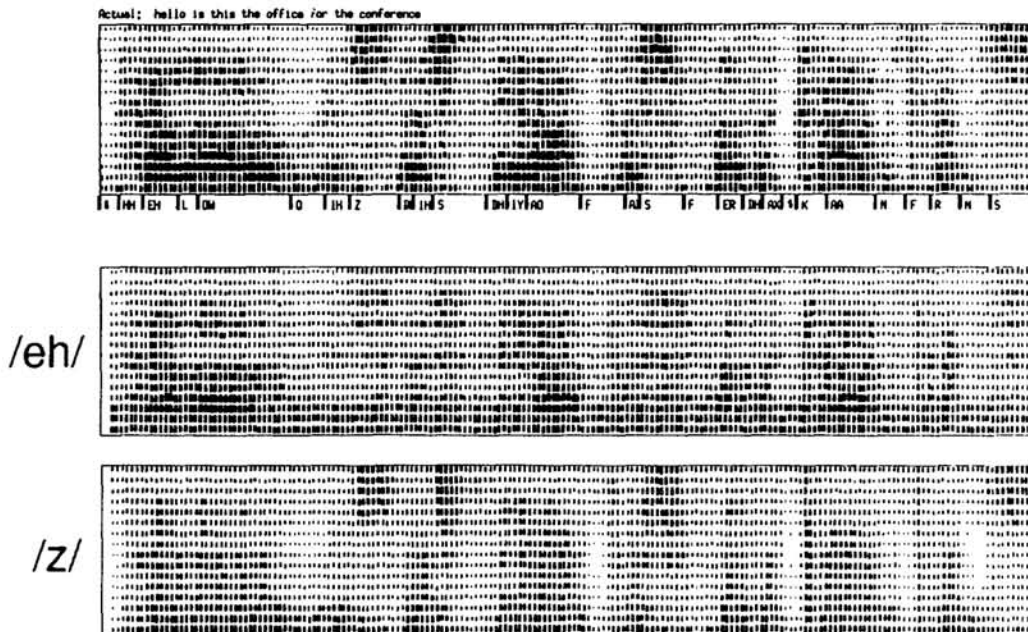

Figure 5: Actual spectrogram, and corresponding predictions by the /eh/ and /z/ phoneme models.

undefined output, which may overlap with the outputs of other predictors. In other words, the predictors are currently only trained on positive instances, because it is not obvious what predictive output target is meaningful for negative instances; and this leads to problematic "undefined regions" for the predictors. Clearly some type of discriminatory training technique should be introduced, to yield better performance in prediction based recognizers.

## 6    CONCLUSION

We have studied the performance of Linked Predictive Neural Networks for large vocabulary, continuous speech recognition. Using a 6-state phoneme topology, without duration modeling or other optimizations, the LPNN achieved an average of 95%, 58%, and 39% accuracy on tasks with perplexity 5, 111, and 402, respectively. This was better than the performance of several simple HMMs that we tested. Further experiments revealed that the accuracy and speed of the LPNN can be slightly improved by the judicious use of hidden control inputs.

The main advantages of predictive networks are that they produce non-binary distortion measures in a simple and elegant way, and that by virtue of their nonlinearity they can model the dynamic properties of speech (e.g., curvature) better than linear predictive models [13]. Their main current weakness is that they have poor discrimination, since their strictly positive training causes them all to make confusably accurate predictions in any context. Future research should concentrate on improving the discriminatory power of the LPNN, by such techniques as corrective training, explicit context dependent phoneme modeling, and function word modeling.

## Acknowledgements

The authors gratefully acknowledge the support of DARPA, the National Science Foundation, ATR Interpreting Telephony Research Laboratories, and NEC Corporation. B. Petek also acknowledges support from the University of Ljubljana and the Research Council of Slovenia. O. Schmidbauer acknowledges support from his employer, Siemens AG, Germany.

# References

[1] H. Bourlard and C. J. Wellekens. Links Between Markov Models and Multilayer Perceptrons. *Pattern Analysis and Machine Intelligence*, 12:12, December 1990.

[2] A. Waibel, T. Hanazawa, G. Hinton, K. Shikano, and K. Lang. Phoneme Recognition Using Time-Delay Neural Networks. *IEEE Transactions on Acoustics, Speech, and Signal Processing*, March 1989.

[3] M. Miyatake, H. Sawai, and K. Shikano. Integrated Training for Spotting Japanese Phonemes Using Large Phonemic Time-Delay Neural Networks. In *Proc. IEEE International Conference on Acoustics, Speech, and Signal Processing*, April 1990.

[4] E. McDermott and S. Katagiri. Shift-Invariant, Multi-Category Phoneme Recognition using Kohonen's LVQ2. In *Proc. IEEE International Conference on Acoustics, Speech, and Signal Processing*, May 1989.

[5] P. Haffner, M. Franzini, and A. Waibel. Integrating Time Alignment and Connectionist Networks for High Performance Continuous Speech Recognition. In *Proc. IEEE International Conference on Acoustics, Speech, and Signal Processing*, May 1991.

[6] K. Iso and T. Watanabe. Speaker-Independent Word Recognition Using a Neural Prediction Model. In *Proc. IEEE International Conference on Acoustics, Speech, and Signal Processing*, April 1990.

[7] E. Levin. Speech Recognition Using Hidden Control Neural Network Architecture. In *Proc. IEEE International Conference on Acoustics, Speech and Signal Processing*, April 1990.

[8] J. Tebelskis and A. Waibel. Large Vocabulary Recognition Using Linked Predictive Neural Networks. In *Proc. IEEE International Conference on Acoustics, Speech, and Signal Processing*, April 1990.

[9] J. Tebelskis, A. Waibel, B. Petek, and O. Schmidbauer. Continuous Speech Recognition Using Linked Predictive Neural Networks. In *Proc. IEEE International Conference on Acoustics, Speech, and Signal Processing*, May 1991.

[10] A. Waibel, A. Jain, A. McNair, H. Saito, A. Hauptmann, and J. Tebelskis. A Speech-to-Speech Translation System Using Connectionist and Symbolic Processing Strategies. In *Proc. IEEE International Conference on Acoustics, Speech, and Signal Processing*, May 1991.

[11] H. Ney. The Use of a One-Stage Dynamic Programming Algorithm for Connected Word Recognition. *IEEE Transactions on Acoustics, Speech, and Signal Processing*, 32:2, April 1984.

[12] H. Ney, A. Noll. Phoneme Modeling Using Continuous Mixture Densities. In *Proc. IEEE International Conference on Acoustics, Speech, and Signal Processing*, April 1988.

[13] N. Tishby. A Dynamic Systems Approach to Speech Processing. In *Proc. IEEE International Conference on Acoustics, Speech, and Signal Processing*, April 1990.